# A Hierarchical Compositional System for Rapid Object Detection

**Long Zhu and Alan Yuille**
Department of Statistics
University of California at Los Angeles
Los Angeles, CA 90095
{lzhu,yuille}@stat.ucla.edu

## Abstract

We describe a hierarchical compositional system for detecting deformable objects in images. Objects are represented by graphical models. The algorithm uses a hierarchical tree where the root of the tree corresponds to the full object and lower-level elements of the tree correspond to simpler features. The algorithm proceeds by passing simple messages up and down the tree. The method works rapidly, in under a second, on $320 \times 240$ images. We demonstrate the approach on detecting cats, horses, and hands. The method works in the presence of background clutter and occlusions. Our approach is contrasted with more traditional methods such as dynamic programming and belief propagation.

## 1  Introduction

Detecting objects rapidly in images is very important. There has recently been great progress in detecting objects with limited appearance variability, such as faces and text [1,2,3]. The use of the SIFT operator also enables rapid detection of rigid objects [4]. The detection of such objects can be performed in under a second even in very large images which makes real time applications practical, see [3].

There has been less progress for the rapid detection of deformable objects, such as hands, horses, and cats. Such objects can be represented compactly by graphical models, see [5,6,7,8], but their variations in shape and appearance makes searching for them considerably harder.

Recent work has included the use of dynamic programming [5,6] and belief propagation [7,8] to perform inference on these graphical models by searching over different spatial configurations. These algorithms are successful at detecting objects but pruning was required to obtain reasonable convergence rates [5,7,8]. Even so, algorithms can take minutes to converge on images of size $320 \times 240$.

In this paper, we propose an alternative methods for performing inference on graphical models of deformable objects. Our approach is based on representing objects in a probabilistic compositional hierarchical tree structure. This structure enables rapid detection of objects by passing messages up and down the tree structure. Our approach is fast with a typical speed of 0.6 seconds on a $320 \times 240$ image (without optimized code).

Our approach can be applied to detect any object that can be represented by a graphical model. This includes the models mentioned above [5,6,7,8], compositional models [9], constellation models [10], models using chamfer matching [11] and models using deformable blur filters [12].

## 2   Background

Graphical models give an attractive framework for modeling object detection problems in computer vision. We use the models and notation described in [8].

The positions of feature points on the object are represented by $\{x_i : i \in \Lambda\}$. We augment this representation to include attributes of the points and obtain a representation $\{q_i : i \in \Lambda\}$. These attributes can be used to model the appearance of the features in the image. For example, a feature point can be associated with an oriented intensity edge and $q_i$ can represent the orientation [8]. Alternatively, the attribute could represent the output of a blurred edge filter [12], or the appearance properties of a constellation model part [10].

There is a prior probability distribution on the configuration of the model $P(\{q_i\})$ and a likelihood function for generating the image data $P(D|\{q_i\})$. We use the same likelihood model as [8]. Our priors are similar to [5,8,12], being based on deformations away from a prototype template.

Inference consists of maximizing the posterior $P(\{q_i\}|D) = P(D|\{q_i\})P(\{q_i\})/P(D)$. As described in [8], this corresponds to a maximizing a posterior of form:

$$P(\{q_i\}|D) = \frac{1}{Z} \prod_i \psi_i(q_i) \prod_{i,j} \psi_{ij}(q_i, q_j), \tag{1}$$

where $\{\psi_i(q_i)\}$ and $\{\psi_{ij}(q_i, q_j)\}$ are the unary and pairwise potentials of the graph. The unary potentials model how well the individual features match to positions in the image. The binary potentials impose (probabilistic) constraints about the spatial relationships between feature points.

Algorithms such as dynamic programming [5,6] and belief propagation [7,8] have been used to search for optima of $P(\{q_i\}|D)$. But the algorithms are time consuming because each state variable $q_i$ can take a large number of values (each feature point on the template can, in principle, match any point in the $240 \times 320$ image). Pruning and other ingenious techniques are used to speed up the search [5,7,8]. But performance remains at speeds of seconds to minutes.

## 3   The Hierarchical Compositional System

We define a compositional hierarchy by breaking down the representation $\{q_i : i \in \Lambda\}$ into substructures which have their own probability models.

At the first level, we group elements into $K_1$ subsets $\{q_i : i \in S_a^1\}$ where $\Lambda = \cup_{a=1}^{K_1} S_a^1$, $S_a^1 \cap S_b^1 = \emptyset$, $a \neq b$. These subsets correspond to meaningful parts of the object, such as ears and other features. See figure (1) for the basic structure. Specific examples for cats and horses will be given later.

For each of these subsets we define a generative model $P_a(D|\{q_i : i \in S_a^1\})$ and a prior $P_a(\{q_i : i \in S_a^1\})$. These generative and prior models are inherited from the full model, see equation (1), by simply cutting the connections between the subset $S_a^1$ and the $\Lambda/S_a^1$ (the remaining features on the object). Hence

$$P_{a^1}(D|\{q_i : i \in S_a^1\}) = \frac{1}{Z_{a^1}} \prod_{i \in S_a^1} \psi_i(q_i)$$

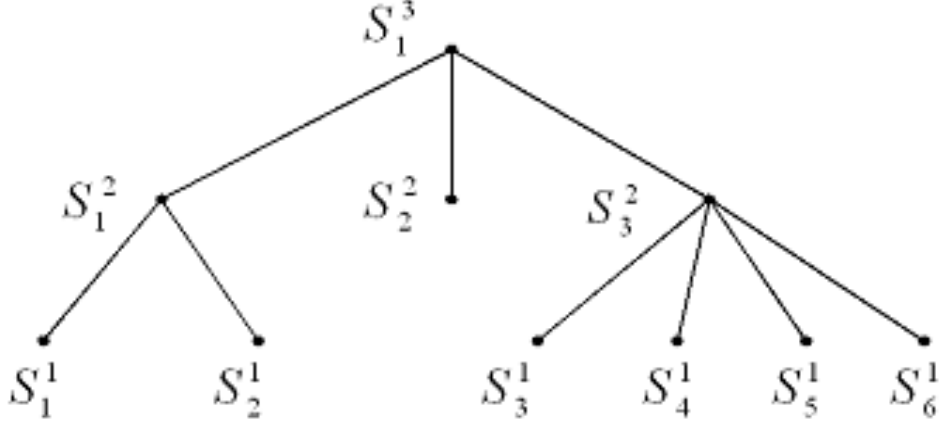

Figure 1: The Hierarchical Compositional structure. The full model contains all the nodes $S_1^3$. This is decomposed into subsets $S_1^2, S_2^2, S_3^2$ corresponding to sub-features. These, in turn, can be decomposed into subsets corresponding to more elementary features.

$$P_{a^1}(\{q_i : i \in S_a^1\}) \quad = \quad \frac{1}{\hat{Z}_{a^1}} \prod_{i,j \in S_a^1} \psi_{ij}(q_i, q_j). \qquad (2)$$

We repeat the same process at the second and higher levels. The subsets $\{S_a^1 : a = 1, ..., K_1\}$ are composed to form a smaller selection of subsets $\{S_b^2 : b = 1, ..., K_2\}$, so that $\Lambda = \cup_{a=1}^{K_2} S_a^2$, $S_a^2 \cap S_b^2 = \emptyset$, $a \neq b$ and each $S_a^1$ is contained entirely inside one $S_b^2$. Again the $S_b^2$ are selected to correspond to meaningful parts of the object. Their generative models and prior distributions are again obtained from the full model, see equation (1). by cutting them off the links to the remaining nodes $\Lambda/S_b^2$.

The algorithm is run using two thresholds $T_1, T_2$. For each subset, say $S_a^1$, we define the *evidence* to be $P_{a^1}(D|\{z_i^\mu : i \in S_a^1\})P_{a^1}(\{z_i^\mu : i \in S_a^1\})$. We determine all possible configurations $\{z_i^\mu : i \in S_a^1\}$ such that evidence of each configuration is above $T_1$. This gives a (possibly large) set of positions for the $\{q_i : i \in S_a^1\}$. We apply non-maximum suppression to reduce many similar configurations in same local area to the one with maximum evidence (measured locally). We observe that a little displacement of position does not change optimality much for upper level matching. Typically, non-maximum suppression keeps around $30 \sim 500$ candidate configurations for each node. These remaining configurations can be considered as *proposals* [13] and are passed up the tree to the subset $S_b^2$ which contains $S_a^1$. Node $S_b^2$ evaluates the proposals to determine which ones are consistent, thus detecting composites of the subfeatures.

There is also top-down message passing which occurs when one part of a node $S_b^2$ contains high evidence – e.g. $P_{a^1}(D|\{z_i^\mu : i \in S_a^1\})P_{a^1}(\{z_i^\mu : i \in S_a^1\}) > T_2$ – but the other child nodes have no consistent values. In this case, we allow the matching to proceed if the combined matching strength is above threshold $T_1$. This mechanism enables the high-level models and, in particular, the priors for the relative positions of the sub-nodes to overcome weak local evidence. This performs a similar function to Coughlan and Shen's dynamic quantization scheme [8].

More sophisticated versions of this approach can be considered. For example, we could use the proposals to activate a data driven Monte Carlo Markov Chain (DDMCMC) algorithm [13]. To our knowledge, the use of hierarchical proposals of this type is unknown in the Monte Carlo sampling literature.

# 4 Experimental Results

We illustrate our hierarchical compositional system on examples of cats, horses, and hands. The images include background clutter and the objects can be partially occluded.

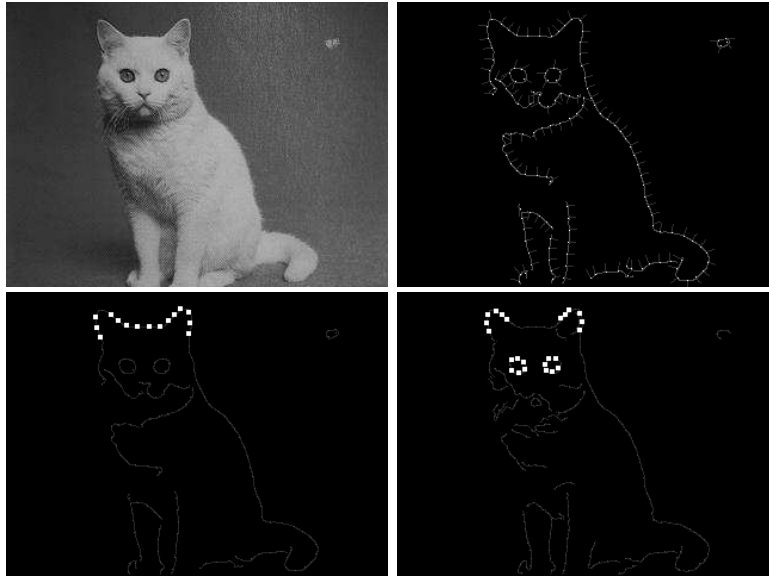

Figure 2: The prototype cat (top left panel), edges after grouping (top right panel), prototype template for ears and top of head (bottom left panel), and prototype for ears and eyes (bottom right panel). 15 points are used for the ears and 24 for the head.

First we preprocess the image using a Canny edge detector followed by simple edge grouping which eliminates isolated edges. Edge detection and edge grouping is illustrated in the top panels of figure (2). This figure is used to construct a prototype template for the ears, eyes, and head – see bottom panels of figure (2).

We construct a graphical model for the cat as described in section (2). Then we define a hierarchical structure, see figure (3).

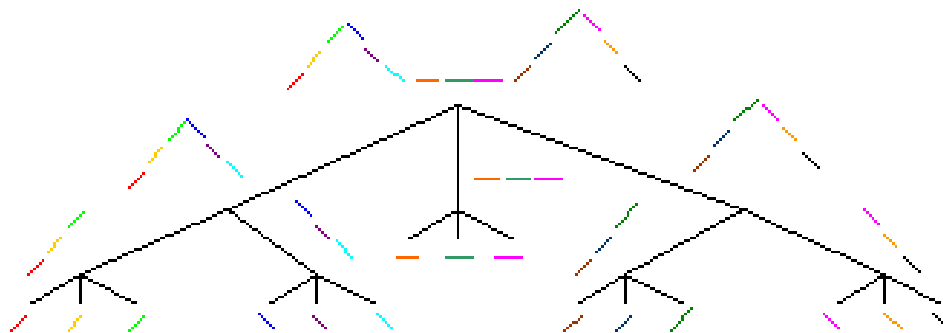

Figure 3: Hierarchy Structure for Cat Template.

Next we illustrate the results on several cat images, see figure (4). Several of these images were used in [8] and we thank Coughlan and Shen for supplying them. In all examples, our

algorithm detects the cat correctly despite the deformations of the cat from the prototype, see figure (2). The detection was performed in less than 0.6 seconds (with unoptimized code). The images are $320 \times 240$ and the preprocessing time is included.

The algorithm is efficient since the subfeatures give bottom-up proposals which constraint the positions of the full model. For example, figure (5) shows the proposals for ears for the cluttered cat image (center panel of figure (4).

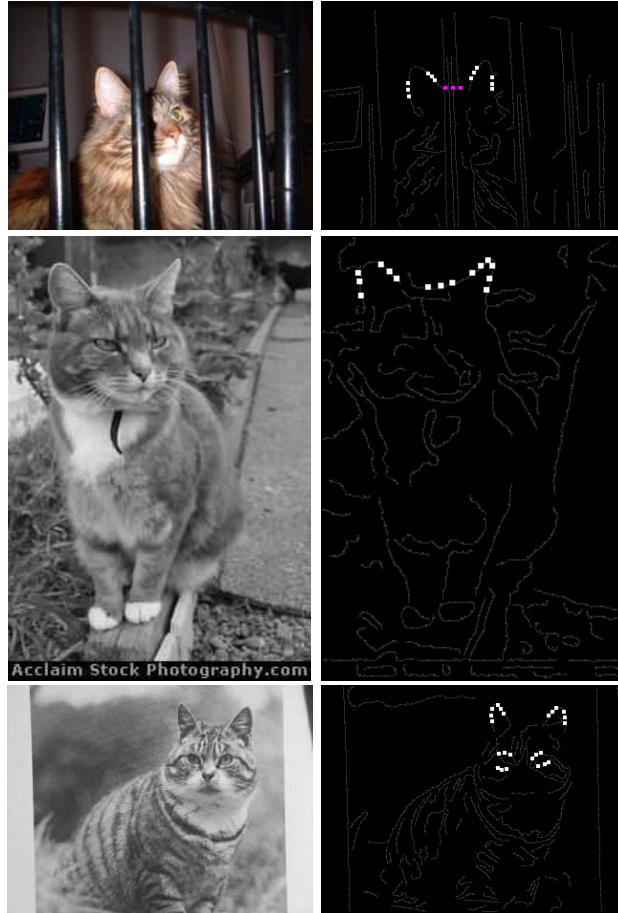

Figure 4: Cat with Occlusion (top panels). Cat with clutter (centre panel). Cat with eyes (bottom panel).

We next illustrate our approach on the tasks of detecting horses. This requires a more complicated hierarchy, see figure (6).

The algorithm succeeds in detecting the horse, see right panels of figure (7), using the prototype template shown in the left panel of figure (7).

Finally, we illustrate this approach for the much studied task of detecting hands, see [5,11]. Our approach detects hand from the Cambridge dataset in under a second, see figure (8). (We are grateful to Thayananthan, Stenger, Torr, and Cipolla for supplying these images).

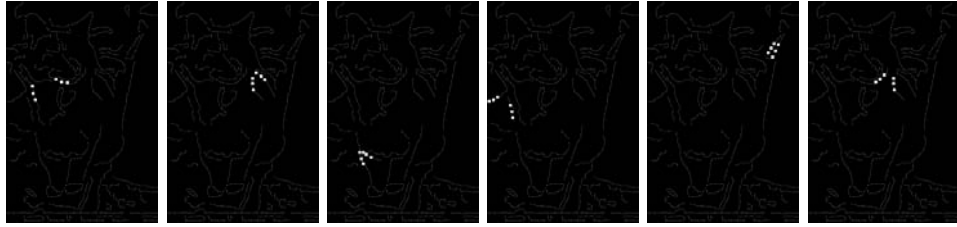

Figure 5: Cat Proposals: Left ears (left three panels). Right ears (right three panels).

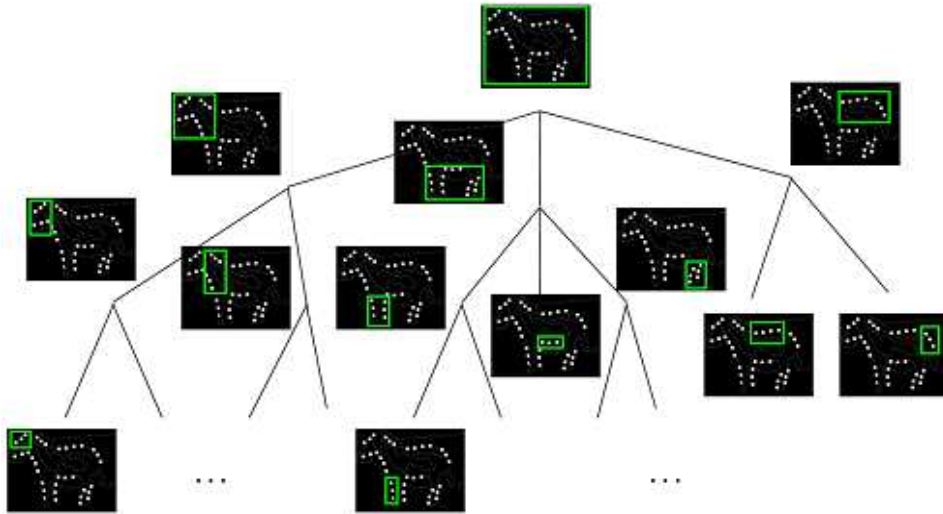

Figure 6: Horse Hierarchy. This is more complicated than the cat.

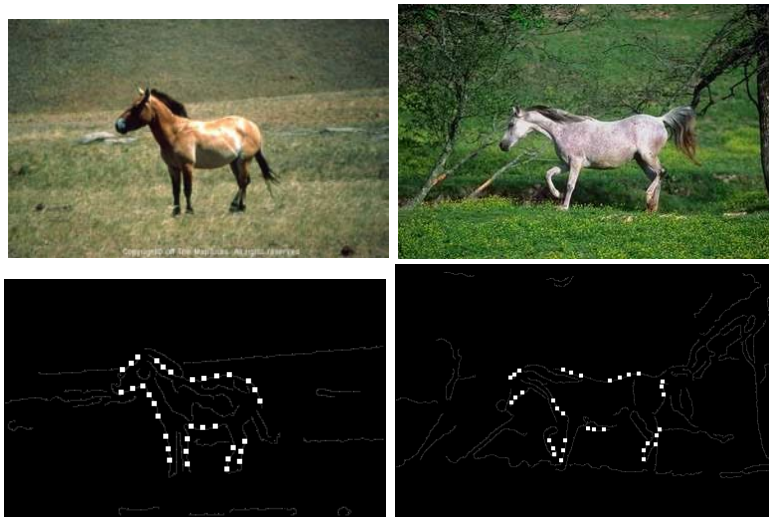

Figure 7: The left panels show the prototype horse (top left panel) and its feature points (bottom left panel). The right panel shows the input image (top right panel) and the position of the horse as detected by the algorithm (bottom right panel).

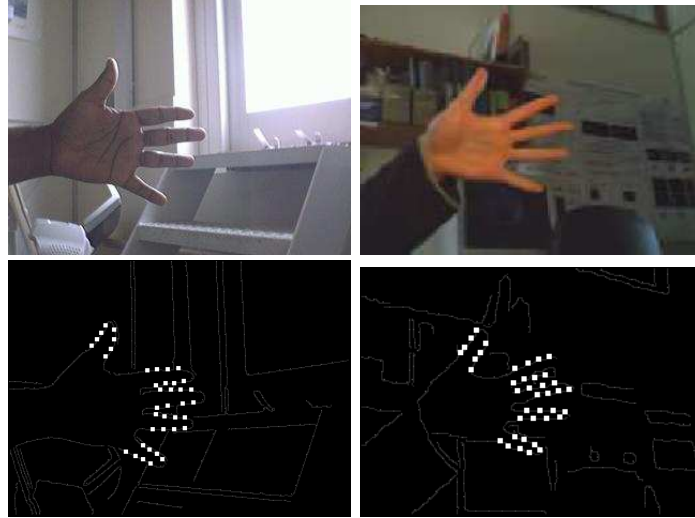

Figure 8: Prototype hand (top left panel), edge map of prototype hand (bottom left panel), Test hand (top right panel), Test hand edges (bottom right panel). 40 points are used.

## 5    Comparison with alternative methods

We ran the algorithm on image of typical size $320 \times 240$. There were usually $4000$ segments after edge grouping. The templates had between 15 and 24 points. The average speed was $0.6$ seconds on a laptop with 1.6 G Intel Pentium CPU (including all processing: edge detector, edge grouping, and object detection.

Other papers report times of seconds to minutes for detecting deformable objects from similar images [5,6,7,8]. So our approach is up to 100 times faster.

The Soft-Assign method in [15] has the ability to deal with objects with around 200 key points, but requires the initialization of the template to be close to the target object. This requirement is not practical in many applications. In our proposed method, there is no need to initialize the template near to the target.

Our hierarchical compositional tree structure is similar to the standard divide and conquer strategy used in some computer science algorithms. This may roughly be expected to scale as $\log N$ where $N$ is the number of points on the deformable template. But precise complexity convergence results are difficult to obtain because they depend on the topology of the template, the amount of clutter in the background, and other factors.

This approach can be applied to any graphical model such as [10,12]. It is straightforward to design hierarchial compositional structures for objects based on their natural decompositions into parts.

There are alternative, and more sophisticated ways, to perform inference on graphical models by decomposing them into sub-graphs, see for example [14]. But these are typically far more computationally demanding.

## 6    Conclusion

We have presented a hierarchical compositional system for rapidly detecting deformable objects in images by performing inference on graphical models. Computation is performed

by passing messages up and down the tree. The systems detects objects in under a second on images of size $320 \times 240$. This makes the approach practical for real world applications.

Our approach is similar in spirit to DDMCMC [13] in that we use proposals to guide the search for objects. In this paper, the proposals are based on a hierarchy of features which enables efficient computation. The low-level features propose more complex features which are validated by the probability models of the complex features. We have not found it necessary to perform stochastic sampling, though it is straightforward to do so in this framework.

## Acknowledgments

This research was supported by NSF grant 0413214.

## References

[1] Viola, P. and Jones, M. (2001). "Fast and Robust Classification using Asymmetric AdaBoost and a Detector Cascade". In *Proceedings NIPS01*.

[2] Schniederman, H. and Kanade, T. (2000). "A Statistical method for 3D object detection applied to faces and cars". In *Computer Vision and Pattern Recognition*.

[3] Chen, X. and Yuille, A.L. (2004). AdaBoost Learning for Detecting and Reading Text in City Scenes. *Proceedings CVPR*.

[4] Lowe, D.G. (1999). "Object recognition from local scale-invariant features." In *Proc. International Conference on Computer Vision ICCV*. Corfu, pages 1150-1157.

[5] Coughlan, J.M., Snow, D., English, C. and Yuille, A.L. (2000). "Efficient Deformable Template Detection and Localization without User Initialization". Computer Vision and Image Understanding. 78, pp 303-319.

[6] Felzenswalb, P. (2005). "Representation and Detection of Deformable Shapes". *IEEE Transactions on Pattern Analysis and Machine Intelligence*, Vol. 27, No. 2.

[7] Coughlan, J.M., and Ferreira, S. (2002). "Finding Deformable Shapes using Loopy Belief Propoagation". In *Proceedings European Conference of Computer Vision.*. 2002.

[8] Coughlan, J.M., and Shen, H. (2004). "Shape Matching with Belief Propagation: Using Dynamic Quantization to Accomodate Occlusion and Clutter". In GMBV .

[9] Geman, S. Potter, D. and Chi, Z. (2002). " Composition systems". Quarterly of Applied Mathematics, LX, pp 707-736.

[10] Fergus, R., Perona, P. and Zisserman, A. (2003) "Object Class Recognition by Unsupervised Scale-Invariant Learning". *Proceeding CVPR*. (2), pp 264-271.

[11] Thayananthan, A. Stenger, B., Torr, P. and Cipolla, R. (2003). "Shape context and chamfer matching in cluttered scenes," In *Proc. Conf. Comp. Vision Pattern Rec.*, pp. 127–133.

[12] Berg, A.C., Berg, T.L., and Malik, J. (2005). "Shape Matching and Object Recognition using Low Distortion Correspondence". *Proceedings CVPR*.

[13] Tu, Z., Chen, X., Yuille, A.L., and Zhu, S.C. (2003). "Image Parsing: Unifying Segmentation, Detection, and Recognition". In *Proceedings ICCV*.

[14] Wainwright, M.J., Jaakkola, T.S., and Willsky., A.S. "Tree-Based Reparamterization Framework for Analysis of Sum-Product and Related Algorithms". *IEEE Transactions on Information Theory*. Vol. 49, pp 1120-1146. No. 5. 2003.

[15] Chui,H. and Rangarajan, A., A New Algorithm for Non-Rigid Point Matching. In Proceedings CVPR 2000.
